# Context-Dependent Classes in a Hybrid Recurrent Network-HMM Speech Recognition System

**Dan Kershaw    Tony Robinson    Mike Hochberg** *

Cambridge University Engineering Department,
Trumpington Street, Cambridge CB2 1PZ, England.
Tel: [+44] 1223 332800, Fax: [+44] 1223 332662.
Email: djk, ajr @eng.cam.ac.uk

## Abstract

A method for incorporating context-dependent phone classes in a connectionist-HMM hybrid speech recognition system is introduced. A modular approach is adopted, where single-layer networks discriminate between different context classes given the phone class and the acoustic data. The context networks are combined with a context-independent (CI) network to generate context-dependent (CD) phone probability estimates. Experiments show an average reduction in word error rate of 16% and 13% from the CI system on ARPA 5,000 word and SQALE 20,000 word tasks respectively. Due to improved modelling, the decoding speed of the CD system is more than twice as fast as the CI system.

## INTRODUCTION

The ABBOT hybrid connectionist-HMM system performed competitively with many conventional hidden Markov model (HMM) systems in the 1994 ARPA evaluations of speech recognition systems (Hochberg, Cook, Renals, Robinson & Schechtman 1995). This hybrid framework is attractive because it is compact, having far fewer parameters than conventional HMM systems, whilst also providing the discriminative powers of a connectionist architecture.

It is well established that particular phones vary acoustically when they occur in different phonetic contexts. For example a vowel may become nasalized when following a nasal sound. The short-term contextual influence of co-articulation is

handled in HMMs by creating a model for all sufficiently differing phonetic contexts with enough acoustic evidence. This modelling of phones in their particular phonetic contexts produces sharper probability density functions. This approach vastly improves HMM recognition accuracy over equivalent context-independent systems (Lee 1989). Although the recurrent neural network (RNN) model acoustic context internally (within the state vector), it does not model phonetic context. This paper presents an approach to improving the ABBOT system through phonetic context-dependent modelling.

In Cohen, Franco, Morgan, Rumelhart & Abrash (1992) separate sets of context-dependent output layers are used to model context effects in different states of HMM phone models. A set of networks discriminate between phones in 8 different broad-class left and right contexts. Training time is reduced by initialising from a CI multi-layer perceptron (MLP) and only changing the hidden-to-output weights during context-dependent training. This system performs well on the DARPA Resource Management Task. The work presented in Zhoa, Schwartz, Sroka & Makhoul (1995) followed along similar work to Cohen et al. (1992). A context-dependent mixture of experts (ME) system (Jordan & Jacobs 1994) based on the structure of the context-independent ME was built. For each state, the whole training data was divided into 46 parts according to its left or right context. Then, a separate ME model was built for each context.

Another approach to phonetic context-dependent modelling with MLPs was proposed by Bourlard & Morgan (1993). It was based on factoring the conditional probability of a phone-in-context given the data in terms of the phone given the data, and its context given the data and the phone. The approach taken in this paper is a mixture of the above work. However, this work augments a recurrent network (rather than an MLP) and concentrates on building a more compact system, which is more suited to our requirements. As a result, the context training scheme is fast and is implemented on a workstation (rather than a parallel processing machine as is used for training the RNN).

## OVERVIEW OF THE ABBOT HYBRID SYSTEM

The basic framework of the ABBOT system is similar to the one described in Bourlard & Morgan (1994) except that a recurrent network is used as the acoustic model for the within the HMM framework. A more detailed description of the recurrent network for phone probability estimation is given in Robinson (1994). At each 16ms time frame, the acoustic vector $\mathbf{u}(t)$ is mapped to an output vector $\mathbf{y}(t)$, which represents an estimate of the posterior probability of each of the phone classes

$$y_i(t) \simeq \Pr(q_i(t)|\mathbf{u}_1^{t+4}), \tag{1}$$

where $q_i(t)$ is phone class $i$ at time $t$, and $\mathbf{u}_1^t = \{\mathbf{u}(1), \ldots, \mathbf{u}(t)\}$ is the input from time 1 to $t$. Left (past) acoustic context is modelled internally by a 256 dimensional state vector $\mathbf{x}(t)$, which can be envisaged as "storing" the information that has been presented at the input. Right (future) acoustic context is given by delaying the posterior probability estimation until four frames of input have been seen by the network. The network is trained using a modified version of error back-propagation through time (Robinson 1994).

Decoding with the hybrid connectionist-HMM approach is equivalent to conventional HMM decoding, with the difference being that the RNN models the state observations. Like typical HMM systems, the recognition process is expressed as finding the maximum *a posteriori* state sequence for the utterance. The decoding criterion specified above requires the computation of the likelihood of the acoustic

data given a phone (state) sequence,

$$p(\mathbf{u}(t)|q_i(t)) = \frac{\Pr(q_i(t)|\mathbf{u}(t))p(\mathbf{u}(t))}{\Pr(q_i)}, \qquad (2)$$

where $p(\mathbf{u}(t))$ is the same for all phones, and hence drops out of the decoding process. Hence, the network outputs are mapped to scaled likelihoods by,

$$p(\mathbf{u}(t)|q_i(t)) \simeq \frac{y_i(t)}{\Pr(q_i)}, \qquad (3)$$

where the priors $\Pr(q_i)$ are estimated from the training data. Decoding uses the NOWAY decoder (Renals & Hochberg 1995) to compute the utterance model that is most likely to have generated the observed speech signal.

## CONTEXT-DEPENDENT PROBABILITY ESTIMATION

The approach taken by this work is to augment the CI RNN, in a similar vein to Bourlard & Morgan (1993). The context-dependent likelihood, $p(\mathbf{U}_t|\mathbf{C}_t, \mathbf{Q}_t)$, can be factored as,

$$p(\mathbf{U}_t|\mathbf{C}_t, \mathbf{Q}_t) = \frac{\Pr(\mathbf{C}_t|\mathbf{U}_t, \mathbf{Q}_t)p(\mathbf{U}_t|\mathbf{Q}_t)}{\Pr(\mathbf{C}_t|\mathbf{Q}_t)}, \qquad (4)$$

where $\mathbf{C}$ is a set of context classes and $\mathbf{Q}$ is a set of context-independent phones or monophones. Substituting for the context independent probability density function, $p(\mathbf{U}_t|\mathbf{Q}_t)$, using (2), this becomes

$$p(\mathbf{U}_t|\mathbf{C}_t, \mathbf{Q}_t) = \frac{\Pr(\mathbf{C}_t|\mathbf{U}_t, \mathbf{Q}_t)\Pr(\mathbf{Q}_t|\mathbf{U}_t)}{\Pr(\mathbf{C}_t|\mathbf{Q}_t)\Pr(\mathbf{Q}_t)}p(\mathbf{U}_t). \qquad (5)$$

The term $p(\mathbf{U}_t)$ is constant for all frames, so this drops out of the decoding process and is ignored for all further purposes. This format is extremely appealing since $\Pr(\mathbf{C}_t|\mathbf{Q}_t)$ and $\Pr(\mathbf{Q}_t)$ are estimated from the training data and the CI RNN estimates $\Pr(\mathbf{Q}_t|\mathbf{U}_t)$. All that is then needed is an estimate of $\Pr(\mathbf{C}_t|\mathbf{U}_t, \mathbf{Q}_t)$. The approach taken in this paper uses a set of *context experts* or *modules* for each monophone class to augment the existing CI RNN.

## TRAINING ON THE STATE VECTOR

An estimate of $\Pr(\mathbf{C}_t|\mathbf{U}_t, \mathbf{Q}_t)$ can be obtained by training a recurrent network to discriminate between contexts $c_j(t)$ for phone class $q_i(t)$, such that

$$y_{j|i}(t) \simeq \Pr(c_j(t)|\mathbf{u}_1^{t+4}, q_i(t)), \qquad (6)$$

where $y_{j|i}(t)$ is an estimate of the posterior probability of context class $j$ given phone class $i$. However, training recurrent neural networks in this format would be expensive and difficult. For a recurrent format, the network must contain no discontinuities in the frame-by-frame acoustic input vectors. This implies all recurrent networks for all the phone classes $i$ must be "shown" all the data. Instead, the assumption is made that since the state vector $\mathbf{x} = f(\mathbf{u})$, then

$$\underline{\mathbf{x}(t+4) \text{ is a good representation for } \mathbf{u}_1^{t+4}.}$$

Hence, a single-layer perceptron is trained on the state vectors corresponding to each monophone, $q_i$, to classify the different phonetic context classes. Finally,

the likelihood estimates for the phonetic context class $j$ for phone class $i$ used in decoding are given by,

$$p(\mathbf{u}(t)|c_j(t), q_i(t)) \simeq \frac{\Pr(q_i(t)|\mathbf{u}_1^{t+4})\Pr(c_j(t)|\mathbf{x}(t+4), q_i(t))}{\Pr(c_j(t)|q_i(t))\Pr(q_i(t))},$$

$$\simeq \frac{y_i(t)y_{j|i}(t)}{\Pr(c_j|q_i)\Pr(q_i)}. \tag{7}$$

Embedded training is used to estimate the parameters of the CD networks and the training data is aligned using a Viterbi segmentation. Each context network is trained on a non-overlapping subset of the state vectors generated from all the Viterbi aligned training data. The context networks were trained using the RProp training procedure (Robinson 1994).

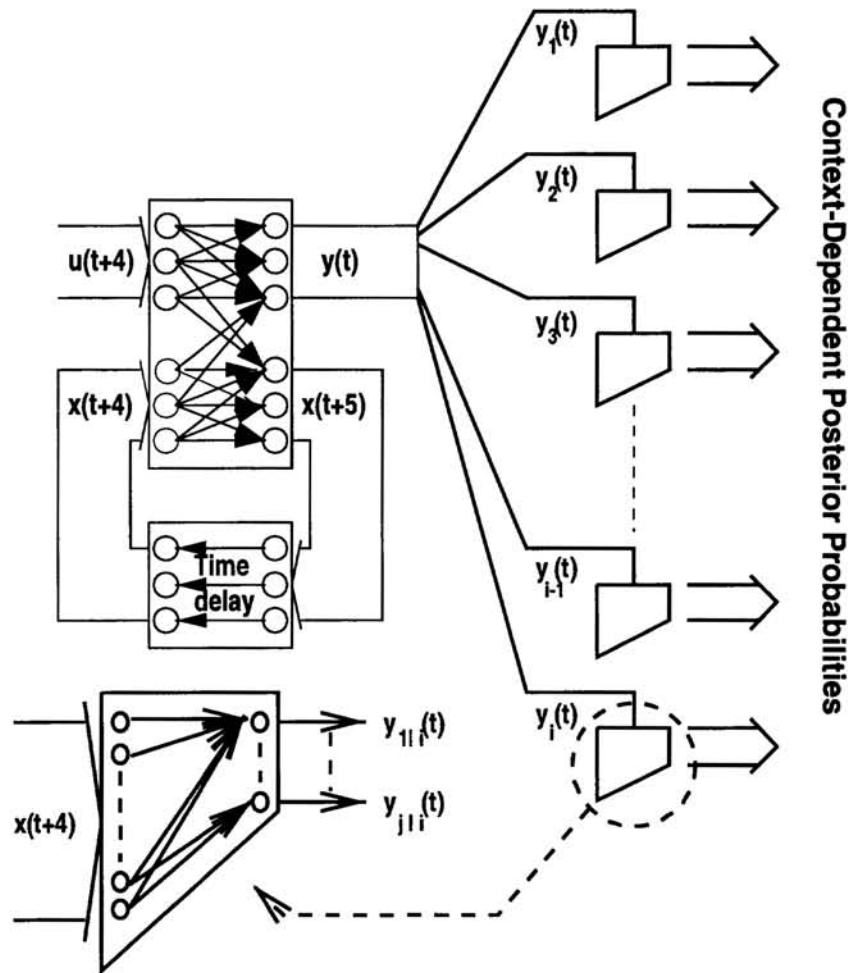

Figure 1: The Phonetic Context-Dependent RNN Modular System.

The frame-by-frame phonetic context posterior probabilities are required as input to the NOWAY decoder, i.e. all the outputs from the context modules on the right hand side of Figure 1. These posterior probabilities are calculated from the numerator of (7). The CI RNN stage operates in its normal fashion, generating frame-by-frame monophone posterior probabilities. At the same time the CD modules take the state vector generated by the RNN as input, in order to classify into a context class. The

RNN posterior probability outputs are multiplied by the module outputs to form context-dependent posterior probability estimates.

## RELATIONSHIP WITH MIXTURE OF EXPERTS

This architecture has similarities with mixture of experts (Jordan & Jacobs 1994). During training, rather than making a "soft" split of the data as in the mixture of experts case, the Viterbi segmentation selects one expert at every exemplar. This means only one expert is responsible for each example in the data. This assumes that the Viterbi segmentation is a good approximation to the segmentation/selection process. Hence, each expert is trained on a small subset of the training data, avoiding the computationally expensive requirement for each expert to "see" all the data. During decoding, the RNN is treated as a gating network, smoothing the predictions of the experts, in an analogous manner to a standard mixture of experts gating network. For further description of the system see Kershaw, Hochberg & Robinson (1995).

## CLUSTERING CONTEXT CLASSES

One of the problems faced by having a context-dependent system is to decide which context classes are to be included in the CD system. A method for overcoming this problem is a decision-tree based approach to cluster the context classes. This guarantees a full coverage of all phones in any context with the context classes being chosen using the acoustic evidence available. The tree clustering framework also allows for the building of a small number of context-dependent phones, keeping the new context-dependent connectionist system architecture compact. The tree building algorithm was based on Young, Odell & Woodland (1994), and further details can be found in Kershaw et al. (1995). Once the trees were built, they were used to relabel the training data and the pronunciation lexicon.

## EVALUATION OF THE CONTEXT SYSTEM

The context-independent networks were trained on the ARPA Wall Street Journal SI84 Corpus. The phonetic context-dependent classes were clustered on the acoustic data according to the decision tree algorithm. Running the data through a recurrent network in a feed-forward fashion to obtain three million frames with 256 dimensional state vectors took approximately 8 hours on an HP735 workstation. Training all the context-dependent networks on all the training data takes between 4–6 hours (in total) on an HP735 workstation. The context-dependent modules were cross-validated on a development set at the word level.

Results for two context-dependent systems, compared with the context-independent baseline are shown in Table 1, where the 1993 spoke 5 test is used for cross-validation and development purposes.

The context-dependent systems were also applied to larger tasks such as the recent 1995 SQALE (a European multi-language speech recognition evaluation) 20,000 word development and evaluation sets. The American English context-dependent system (CD527) was extended to include a set of modules trained backwards in time (which were log-merged with the forward context), to augment a four way log-merged context-independent system (Hochberg, Cook, Renals & Robinson 1994).

Table 1: Comparison Of The CI System With The CD205 And CD527 Systems, For 5000 Word, Bigram Language Model Tasks.

| 1993 Test Sets | CI System WER | CD205 System | | CD527 System | |
|---|---|---|---|---|---|
| | | WER | Red$^n$ WER | WER | Red$^n$ WER |
| Spoke 5 | 16.0 | 14.0 | 12.7 | 13.6 | 14.9 |
| Spoke 6 | 14.6 | 12.2 | 16.3 | 11.7 | 19.8 |
| Eval. | 15.7 | 14.3 | 8.4 | 13.7 | 12.6 |

Table 2: Comparison Of The Merged CI Systems With The CD527US And CD465UK Systems, For 20,000 Word Tasks. All Tests Use A Trigram Language Model. The CD527US And CD465UK Evaluation Results Have Been Officially Adjudicated.

| 1995 Test Sets | CI System WER | CD System WER | Red$^n$ WER |
|---|---|---|---|
| US English dev_test | 12.8 | 11.3 | 12.2 |
| US English evl_test | 14.5 | 12.9$^\dagger$ | 9.8 |
| UK English dev_test | 15.6 | 12.7 | 18.9 |
| UK English evl_test | 16.4 | 13.8$^\dagger$ | 15.7 |

Table 3: Comparison Of Average Utterance Decode Speed Of The CI Systems With The CD527US And CD465UK Systems On An HP735, For 20,000 Word Tasks. All Tests Use A Trigram Language Model, And The Same Pruning Levels.

| Tests | CI Utterance Av. Decode Speed (s) | CD Utterance Av. Decode Speed (s) | Speedup |
|---|---|---|---|
| American English | 67 | 31 | 2.16 |
| British English | 131 | 48 | 2.73 |

Table 4: The Number Of Parameters Used For The CI Systems As Compared With The CD527US And CD465UK Systems.

| System | # CI Parameters | # CD Parameters | % Increase In Parameters |
|---|---|---|---|
| American English | 341,000 | 612,000 | 79.0 |
| British English | 331,000 | 570,000 | 72.2 |

A similar system was built for British English (CD465). Table 2 shows the improvement gained by using context models. The daggers indicate the official entries for the 1995 SQALE evaluation. These figures represent the lowest reported word error rate for both the US and UK English tasks.

As a result of improved phonetic modelling and class discrimination the search space was reduced. This meant that decoding speed was over twice as fast as the context-dependent system, Table 3, even though there were roughly ten times as many context-dependent phones compared to the monophones.

The increase in the number of parameters due to the introduction of the context models for the SQALE evaluation system are shown in Table 4. Although this seems a large increase in the number of system parameters, it is still an order of magnitude less than any equivalent HMM system built for this task.

## CONCLUSIONS

This paper has discussed a successful way of integrating phonetic context-dependent classes into the current ABBOT hybrid system. The architecture followed a modular approach which could be used to augment any current RNN-HMM hybrid system. Fast training of the context-dependent modules was achieved. Training on all of the SI84 corpus took between 4 and 6 hours. Utterance decoding was performed using the standard NOWAY decoder. The word error was significantly reduced, whilst the decoding speed of the context system was over twice as fast as the baseline system (for 20,000 word tasks).

## Footnotes

*Mike Hochberg is now at Nuance Communications, 333 Ravenswood Avenue, Building 110, Menlo Park, CA 94025, USA. Tel: [+1] 415 6148260.

## References

Bourlard, H. & Morgan, N. (1993), 'Continuous Speech Recognition by Connectionist Statistical Methods', *IEEE Transactions on Neural Networks* 4(6), 893–909.

Bourlard, H. & Morgan, N. (1994), *Connectionist Speech Recognition: A Hybrid Approach*, Kluwer Acedemic Publishers.

Cohen, M., Franco, H., Morgan, N., Rumelhart, D. & Abrash, V. (1992), Context-Dependent Multiple Distribution Phonetic Modeling with MLPs, *in* 'NIPS 5'.

Hochberg, M., Cook, G., Renals, S. & Robinson, A. (1994), Connectionist Model Combination for Large Vocabulary Speech Recognition, *in* 'Neural Networks for Signal Processing', Vol. IV, pp. 269–278.

Hochberg, M., Cook, G., Renals, S., Robinson, A. & Schechtman, R. (1995), The 1994 ABBOT Hybrid Connectionist-HMM Large-Vocabulary Recognition System, *in* 'Spoken Language Systems Technology Workshop', ARPA, pp. 170–6.

Jordan, M. & Jacobs, R. (1994), 'Hierarchical Mixtures of Experts and the EM Algorithm', *Neural Computation* 6, 181–214.

Kershaw, D., Hochberg, M. & Robinson, A. (1995), Incorporating Context-Dependent Classes in a Hybrid Recurrent Network-HMM Speech Recognition System, F-INFENG TR217, Cambridge University Engineering Department.

Lee, K.-F. (1989), *Automatic Speech Recognition; The Development of the SPHINX System*, Kluwer Acedemic Publishers.

Renals, S. & Hochberg, M. (1995), Efficient Search Using Posterior Phone Probability Estimates, *in* 'ICASSP', Vol. 1, pp. 596–9.

Robinson, A. (1994), 'An Application of Recurrent Nets to Phone Probability Estimation.', *IEEE Transactions on Neural Networks* 5(2), 298–305.

Young, S., Odell, J. & Woodland, P. (1994), 'Tree-Based State Tying for High Accuracy Acoustic Modelling', *Spoken Language Systems Technology Workshop*.

Zhoa, Y., Schwartz, R., Sroka, J. & Makhoul, J. (1995), Hierarchical Mixtures of Experts Methodology Applied to Continuous Speech Recognition, *in* 'NIPS 7'.
